# Priors over Recurrent Continuous Time Processes

**Ardavan Saeedi     Alexandre Bouchard-Côté**
Department of Statistics
University of British Columbia

## Abstract

We introduce the Gamma-Exponential Process (GEP), a prior over a large family of continuous time stochastic processes. A hierarchical version of this prior (HGEP; the Hierarchical GEP) yields a useful model for analyzing complex time series. Models based on HGEPs display many attractive properties: conjugacy, exchangeability and closed-form predictive distribution for the waiting times, and exact Gibbs updates for the time scale parameters. After establishing these properties, we show how posterior inference can be carried efficiently using Particle MCMC methods [1]. This yields a MCMC algorithm that can resample entire sequences atomically while avoiding the complications of introducing slice and stick auxiliary variables of the beam sampler [2]. We applied our model to the problem of estimating the disease progression in multiple sclerosis [3], and to RNA evolutionary modeling [4]. In both domains, we found that our model outperformed the standard rate matrix estimation approach.

## 1   Introduction

The application of non-parametric Bayesian techniques to time series has been an active field in the recent years, and has led to many successful continuous time models. Examples include Dependent Dirichlet Processes (DDP) [5], Ornstein-Uhlenbeck Dirichlet Processes [6], and stick-breaking autoregressive processes [7]. One property of these models is that they are *forgetful*, meaning that the effect of an observation at time $t$ on a prediction at time $t + s$ will decrease as $s \to \infty$. More formally, DDPs and their cousins can be viewed as priors over transient processes (see Section A of the Supplementary Material).

In some situations, emphasizing the short term trends is desirable, for example for the analysis of financial time series. However, in other situations, this behavior does not use the data optimally.

As a concrete example of the type of time series we are interested in, consider the problem of modeling the progression of recurrent diseases such as multiple sclerosis. Recurrent diseases are characterized by alternations between relapse and remission periods, and patients can undergo this cycle repeatedly. In multiple sclerosis research, measuring the effect of drugs in the presence of these complex cycles is challenging, and is one of the applications that motivated this work.

The data available to infer the disease progression typically takes the form of summary measurements taken at different points in time for each patient. We model these measurements as being conditionally independent given a continuous time non-parametric latent process. The main options available for this type of situation are currently limited to parametric Bayesian models [8], or to non-Bayesian models [9].

In this work, we propose a family of models, Gamma-Exponential Processes (GEPs), that fills this gap. GEPs are based on priors over recurrent, infinite rate matrices specifying a jump process in a latent space.

It is informative to start by a preview of what the predictive distributions look like in GEP models. Indeed, an advantage of GEPs is that they have simple predictive distributions, a situation remi-

niscent of the theory of Dirichlet Processes, in which the simple predictive distributions (given by the Chinese Restaurant Process (CRP)) were probably an important factor behind their widespread adoption in Bayesian non-parametric statistics.

Suppose that the hidden state at the current time step is $\theta$, and that we are interested in the distribution over the waiting time $t$ before the next jump to a different hidden state (we will come back to the predictive distribution over what this next state is in Section 3, showing that it has the form of a CRP). Let $t_1, t_2, \ldots, t_n$ denote the previous, distinct waiting times at $\theta$. The predictive distribution is then specified by the following density over the positive reals:

$$f(t) = \frac{(\alpha_0 + n)(\beta_0 + T)^{(\alpha_0 + n)}}{(\beta_0 + T + t)^{\alpha_0 + n + 1}},$$

where $T$ is the sum over the $t_i$'s, and $\alpha_0, \beta_0$ are parameters. It can be checked that this yields an exchangeable distribution over the sequences of waiting times at $\theta$ (if forms a telescoping product— see the proof of Proposition 5 in the Supplementary Material). By de Finetti's theorem, there is therefore a mixing prior distribution. We identify this prior in Section 3, and use it to build a powerful hierarchical model in Section 4. As we will see, this hierarchical model displays many attractive properties: conjugacy, exchangeability and closed-form predictive distributions for the waiting times, and exact Gibbs updates for the time scale parameters. Moreover it admits efficient inference algorithms, described in Section 5.

In addition to the connection to DDPs mentioned above, our models are also related to the infinite Hidden Markov Model (iHMM) [10] and to the more general Sticky-HDP-HMM [11], which are both based on priors over *discrete* time processes. While continuous-time analogues of these discrete time processes can be constructed by subordination, we discuss in Section C of the Supplementary Material the differences and advantages of GEPs compared to these subordinations. A similar argument holds for factorial extensions of the infinite HMM [12].

Gamma (Moran) Processes [13], a building block for our process, have been used in non-parametric Bayesian statistics, but in different contexts, for example in survival analysis [14], spatial statistics [15], and for modeling count data [16].[1] Note also that the gamma-exponential process introduced here is unrelated to the exponential-gamma process [18].

## 2   Background and notation

While our process can be defined on continuous state spaces, the essential ideas can be described over countable state spaces. We therefore focus in this section on reviewing Continuous Time Markov Processes (CTMPs) over a countably infinite state space.

These CTMPs can be characterized by an infinite matrix $q_{i,j}$ where the off-diagonal entries are non-negative and each row sums to zero (i.e. the diagonal entries are negative and with magnitude equal to the sum of the off-diagonal row entries). Samples from these processes take the form of a list of pairs of states and waiting times $X = (\theta_n, J_n)_{n=1}^{N}$ (see Figure 1(a)). We will call each pair of that form a (hidden) event. Typically, only a function $\mathcal{Y}$ of the events is available. For example, measurements could be taken at fixed or random time intervals. We will come back to the partially observed sequences setup in Section 5.

To simulate a sequence of events given parameters $Q = (q_{i,j})$, we use the standard Doob-Gillespie algorithm: conditioning on the current state having index $i$, $\theta_N = i$, the waiting time before the next jump is exponentially distributed $J_{N+1}|(\theta_N = i) \sim \mathrm{Exp}(-q_{i,i})$, and the index $j \neq i$ of the next state $\theta_{N+1}$ is selected independently with probability proportional to $p(j) = q_{i,j}\,\mathbf{1}[i \neq j]$.

The goal of this work is to develop priors on such infinite rate matrices that are both flexible and easy to work with. To do that, we first note that the off-diagonal elements of each row $i$ can be viewed as a positive measure $\mu_i$. Note that the normalization of this measure in not equal to one in general. We will denote the normalization constant of measures by $||\mu||$ and the normalized measures by $\bar{\mu} = \mu/||\mu||$.

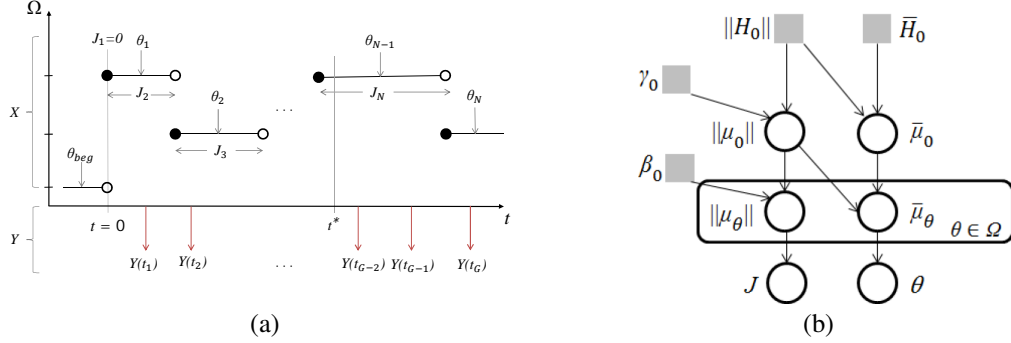

(a)  (b)

Figure 1: (a) An illustration of our notation for samples from CTMPs. We assume the state space ($\Omega$) is countable. The notation for the observations $Y(t_1), \ldots, Y(t_G)$ is described in Section 5. (b) Graphical model for the hierarchical model of Section 4. For simplicity we only show a single $J$ and $\theta$.

To get a conjugate family, we will base our priors on Moran Gamma Processes (MGPs) [13], a family of measure-valued probability distributions. MGPs have three parameters: (1) A positive real number $\alpha_0 > 0$, called the *concentration* or *shape parameter*, (2) A probability distribution $P_0 : \mathcal{F}_\Omega \to [0, 1]$ called the *base probability distribution*, (3) A positive real number $\beta_0 > 0$, called the *rate parameter*. Alternatively, the first two parameters can be grouped into a single finite *base measure parameter* $H_0 = \alpha_0 P_0$.

Recall that by the Kolmogorov consistency theorem, in order to guarantee the existence of a stochastic process on a probability space $(\Omega', \mathcal{F}_{\Omega'})$, it is enough to provide a consistent definition of what the marginals of this stochastic process are. As the name suggest, in the case of a Moran Gamma process, the marginals are gamma distributions:

**Definition 1** (Moran Gamma Process). *Let $H_0, \beta_0$ be of the types listed above. We say that $\mu :$ $\mathcal{F}_{\Omega'} \to (\mathcal{F}_\Omega \to [0, \infty))$ is distributed according to the* Moran Gamma process distribution, *denoted by $\mu \sim \mathrm{MGP}(H_0, \beta_0)$, if for all measurable partitions of $\Omega$, $(A_1, \ldots, A_K)$, we have:* [2]

$$(\mu(A_1), \mu(A_2), \ldots, \mu(A_K)) \sim \mathrm{Gamma}(H_0(A_1), \beta_0) \times \cdots \times \mathrm{Gamma}(H_0(A_k), \beta_0).$$

## 3  Gamma-Exponential Process

We can now describe the basic version of our model, the Gamma-Exponential Process (GEP). In the next section, we will move to a hierarchical version of this model.

In GEPs, the rows of a rate matrix $Q$ are obtained by a transformation of iid samples from an MGP, and the states are then generated from $Q$ with the Doob-Gillespie algorithm described in the previous section. In this section we show that this model is conjugate and has a closed form expression for the predictive distribution.

Let $H_0$ be a base measure on a countable support $\Omega$ with $\|H_0\| < \infty$. We will relax the countable base measure support assumption in the next section. The GEP is formally defined as follows:

$$\mu_\theta \overset{\text{iid}}{\sim} \mathrm{MGP}(H_0, \beta_0) \ \forall \theta \in \Omega$$

$$\theta_{N+1} \big| X, \{\mu_\theta\}_{\theta \in \Omega} \sim \bar{\mu}_{\theta_N}$$

$$J_{N+1} \big| X, \{\mu_\theta\}_{\theta \in \Omega} \sim \mathrm{Exp}\left(\|\mu_{\theta_N}\|\right)$$

To understand the connection with the Doob-Gillespie process, note that a rate matrix can be obtained by arbitrarily ordering $\Omega = \theta^{(1)}, \theta^{(2)}, \ldots$, and setting:[3] $q_{i,j} = \mu_{\theta^{(i)}}(\{\theta^{(j)}\})$ if $i \neq j$, and

$\|\mu_{\theta^{(i)}}\|\ (\bar{\mu}_{\theta^{(i)}}(\{i\})-1)$ otherwise. In order to model the initial distribution without cluttering the notation, we assume there is a special state $\theta_{\text{beg}}$ always present at the beginning of the sequence, and only at the beginning. In other words, we always condition on $(\theta_0 = \theta_{\text{beg}})$ and $(\theta_n \neq \theta_{\text{beg}}, n > 0)$, and drop these conditioning events from the notation. Similarly, we are going to consider distribution over infinite sequences in the notation that follows, but if the goal is to model finite sequences, an additional special state $\theta_{\text{end}} \neq \theta_{\text{beg}}$ can be introduced. We would then condition on $(\theta_{N+1} = \theta_{\text{end}})$ and $(\theta_n \neq \theta_{\text{end}}, n \in \{1, \ldots, N\})$, and set the total rate for the row corresponding to $\theta_{\text{end}}$ to zero.

Next, we show that the posterior of each row, $\mu_\theta | X$, is also MGP distributed with updated parameters. We assume that all the states are observed for now, and treat the partially observed case in Section 5.

The sufficient statistics for the parameters of $\mu_\theta | X$ are the empirical transition measures and waiting times:

$$F_\theta = \sum_{n=1}^{N} \mathbf{1}[\theta_{n-1} = \theta]\, \delta_{\theta_n}, \qquad T_\theta = \sum_{n=1}^{N} \mathbf{1}[\theta_{n-1} = \theta]\, J_n.$$

**Proposition 2.** *The Gamma-Exponential Process (GEP) is a conjugate family, $\mu_\theta | X \sim$ MGP $(\mu'_\theta, \beta'_\theta)$, where $\mu'_\theta = F_\theta + H_0$ and $\beta'_\theta = T_\theta + \beta_0$.*

Note that the $\mu'_\theta$ are unnormalized versions of the posterior parameters of a Dirichlet process. This connexion with the Dirichlet process is used in the proof below, and also implies that samples from GEPs have countable support even when $\Omega$ is uncountable (i.e. the chain will always visit a random countable subset of $\Omega$). For the proof of proposition 2, we will need the following elementary lemma:

**Lemma 3.** *If $V \sim \text{Beta}(a, b)$ and $W \sim \text{Gamma}(a + b, c)$ are independent, then $VW \sim \text{Gamma}(a, c)$.*

See for example [19] for a survey of standard beta-gamma algebra results such as the one stated in this lemma. We now prove the proposition:

*Proof.* Fix an arbitrary state $\theta$ and drop the index for simplicity (this is without loss of generality since the rows are iid): let $\mu = \mu_\theta$, $\mu' = \mu'_\theta$, and $\beta' = \beta'_\theta$.

Let $(A_1, \ldots, A_K)$ be a measurable partition of $\Omega$. By the Kolmogorov consistency theorem, it is enough to show that for all such partition,

$$(\mu(A_1), \mu(A_2), \ldots, \mu(A_K)) | X \sim \text{Gamma}(\mu'(A_1), \beta') \times \cdots \times \text{Gamma}(\mu'(A_k), \beta').$$

Assume for simplicity that $K = 2$ (the argument can be generalized to $K > 2$ without difficulties), and let $\Gamma_1 = \mu(A_1), \Gamma_0 = \|\mu\|$. By elementary properties of Gamma distributed vectors, if we let $V = \Gamma_1/\Gamma_0, W = \Gamma_0$, then $V \sim \text{Beta}(H_0(A_1), H_0(A_2)), W \sim \text{Gamma}(\alpha_0, \beta_0)$, and $V, W$ are independent (both conditionally on $X$ and unconditionally). By beta-multinomial conjugacy, we also have $(V|X) = (V|\theta_1, \ldots, \theta_N) \sim \text{Beta}(\mu'(A_1), \mu'(A_2))$, and by gamma-exponential conjugacy, we have $W|X \sim \text{Gamma}(\|\mu'\|, \beta')$.

Using the lemma with $a = \mu'(A_1), b = \mu'(A_2), c = \beta'$, we finally get that $(\mu(A_1)|X) = (VW|X) \sim \text{Gamma}(\mu'(A_1), \beta')$, which concludes the proof. $\qquad\square$

We now turn to the task of finding an expression for the predictive distribution, $(\theta_{N+1}, J_{N+1})|X$. We will need the following family of densities (see Section F for more information):

**Definition 4** (Translated Pareto). *Let $\alpha > 0, \beta > 0$. We say that a random variable $T$ is* translated-Pareto*, denoted $T \sim \text{TP}(\alpha, \beta)$, if it has density:*

$$f(t) = \frac{\mathbf{1}[t > 0]\alpha\beta^\alpha}{(t + \beta)^{\alpha+1}}. \tag{1}$$

**Proposition 5.** *The predictive distribution of the GEP is given by:*

$$(\theta_{N+1}, J_{N+1})|X \sim \bar{\mu}'_{\theta_N} \times \text{TP}(\|\mu'_{\theta_N}\|, \beta'_{\theta_N}). \tag{2}$$

*Proof.* By Proposition 2, it is enough to show that if $\mu \sim \mathrm{MGP}(H_0, \beta_0)$, $\theta|\mu \sim \bar{\mu}$, and $J|\mu \sim \mathrm{Exp}(\|\mu\|)$, then $(\theta, J) \sim \bar{\mu} \times \mathrm{TP}(\alpha_0, \beta_0)$, where $\alpha_0 = \|H_0\|$.

Note first that we have $(J|\theta) \overset{d}{=} J$ by the fact that the minimum and argmin of independent exponential random variables are independent. To get the distribution of $J$, we need to show that the following integral is proportional to Equation (1):

$$p(t) \propto \int_{x>0} x^{\alpha_0 - 1} \exp(-\beta_0 x) \cdot x \exp(-xt) \, \mathrm{d}x$$

$$= \int_{x>0} x^{\alpha_0} \exp\left(-(\beta_0 + t)x\right) \, \mathrm{d}x = \frac{\Gamma(\alpha_0 + 1)}{(\beta_0 + t)^{\alpha_0 + 1}}$$

Hence $J \sim \mathrm{TP}(\alpha_0, \beta_0)$. $\qquad\square$

As a sanity check, and to connect this result with the discussion in the introduction, it is instructive to directly check that these predictive distributions are indeed exchangeable (see Section B for the proof):

**Proposition 6.** *Let* $J_{j(\theta,1)}, J_{j(\theta,2)}, \ldots, J_{j(\theta,K)}$ *be the subsequence of waiting times following state* $\theta$. *Then the random variables* $J_{j(\theta,1)}, J_{j(\theta,2)}, \ldots, J_{j(\theta,K)}$ *are exchangeable. Moreover, the joint density of a sequence of waiting times* $(J_{j(\theta,1)} = j_1, J_{j(\theta,2)} = j_2, \ldots, J_{j(\theta,K)} = j_K)$ *is given by:*

$$p(j_1, j_2, \ldots, j_K) = \frac{\mathbf{1}[j_k > 0, k \in \{1, \ldots, K\}](\alpha_0)_K \beta_0^{\alpha_0}}{(\beta_0 + j_1 + \cdots + j_K)^{\alpha_0 + K}} \tag{3}$$

*where the Pochhammer symbol* $(x)_n$ *is defined as* $(x)_n = x(x+1) \cdots (x+n-1)$.

# 4 Hierarchical GEP

In this section, we present a hierarchical version of the GEP, where the rows of the random rate matrix are exchangeable rather than iid. Informally, the motivation behind this construction is to have the rows share information on what states are frequently visited.

As with Hierarchical Dirichlet Processes (HDPs) [20], the hierarchical construction is especially important when $\Omega$ is uncountable. For such spaces, since each GEP sample has a random countable support, any two independent GEP samples will have disjoint supports with probability one. Therefore, GEP alone cannot be used to construct recurrent processes when $\Omega$ is uncountable. Fortunately, the hierarchical model introduced in this section addresses this issue: it yields a recurrent prior over continuous time jump processes over both countable and uncountable spaces $\Omega$ (see Section A).

The hierarchical process is constructed by making the base measure parameter of the rows shared and random. Formally, the model has the following form:

$$\begin{cases} \mu_0 & \sim \mathrm{MGP}(H_0, \gamma_0) \\ \mu_\theta|\mu_0 & \overset{\mathrm{iid}}{\sim} \mathrm{MGP}(\mu_0, \beta_0) \end{cases} \qquad \begin{cases} \theta_{N+1}\big|X, \{\mu_\theta\}_{\theta \in \Omega} & \sim \bar{\mu}_{\theta_N} \\ J_{N+1}\big|X, \{\mu_\theta\}_{\theta \in \Omega} & \sim \mathrm{Exp}(\|\mu_{\theta_N}\|). \end{cases}$$

In order to get a tractable predictive distribution, we introduce a set of auxiliary variables. These auxiliary variables can be compared to the variables used in the Chinese Restaurant Franchise (CRF) metaphor [20] to indicate when new *tables* are created in a given *restaurant*. In the HGEP, a restaurant can be understood as a row in the rate matrix, and tables, as groups of transitions to the same destination state. These auxiliary variables will be denoted by $A_n$, where the event $A_n = 1$ means informally that the $n$-th transition creates a new table. The variable takes value $A_n = 0$ otherwise. See Section D in the Supplementary Material for a review of the CRF construction and a formal definition of the auxiliary variables $A_n$.

We augment the sufficient statistics with empirical counts for the number of tables across all restaurants that share a given dish, $G = \sum_{n=1}^{N} A_n \delta_{\theta_n}$, and introduce one additional auxiliary variable, the normalization of the top level random measure, $\|\mu_0\|$. This latter auxiliary variable has no equivalent in CRFs. As in the previous section, the normalization of the lower level random measures $\|\mu_\theta\|$ will be marginalized. Finally, we let:

$$\mu'' = G + H_0 \qquad\qquad \mu_\theta'^{(\mathrm{H})} = F_\theta + \|\mu_0\| \bar{\mu}''$$

where $\bar{\mu}_\theta'^{(\mathrm{H})}$ can be recognized as the mean parameter of the predictive distribution of the HDP. We use the superscript (H) to disambiguate from the non-hierarchical case. The main result of this section is (see Section B for the proof):

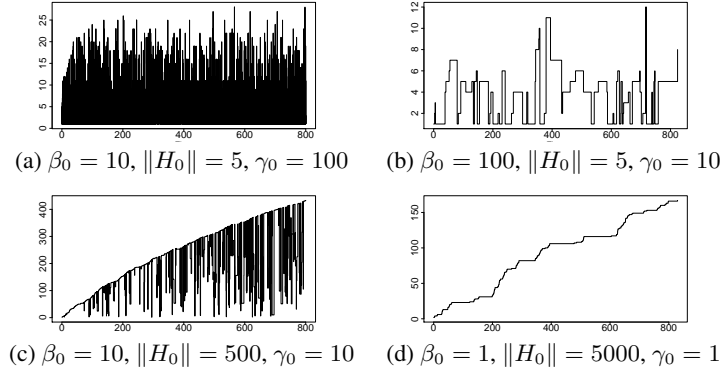

(a) $\beta_0 = 10, \|H_0\| = 5, \gamma_0 = 100$  (b) $\beta_0 = 100, \|H_0\| = 5, \gamma_0 = 10$

(c) $\beta_0 = 10, \|H_0\| = 500, \gamma_0 = 10$  (d) $\beta_0 = 1, \|H_0\| = 5000, \gamma_0 = 1$

Figure 2: Qualitative behavior of the prior

**Proposition 7.** *The predictive distribution of the Hierarchical GEP (HGEP) is given by:*

$$(\theta_{N+1}, J_{N+1}) | (X, \{A_n\}_{n=1}^N, \|\mu_0\|) \sim \bar{\mu}'^{(H)}_{\theta_N} \times \mathrm{TP}(\|\mu'^{(H)}_{\theta_N}\|, \beta'_{\theta_N}).$$

To resample the auxiliary variable $\|\mu_0\|$, a gamma-distributed Gibbs kernel can be used (see Section E of the Supplementary Material).

## 5  Inference on partially observed sequences

In this section, we describe how to approximate expectations under the posterior distribution of GEPs, $\mathbb{E}[h(X)|\mathcal{Y}]$, for a test function $h$ on the hidden events $X$ given observations $\mathcal{Y}$. An example of function $h$ on these events is to interpolate the progression of the disease in a patient with Multiple Sclerosis (MS) between two medical visits. We start by describing the form of the observations $\mathcal{Y}$.

Note that in most applications, the sequence of states is not directly nor fully observed. First, instead of observing the random variables $\theta$, inference is often carried from $\mathcal{X}$-valued random variables $Y_n$ distributed according to a parametric family $\mathcal{P}$ indexed by the states $\theta$ of the chain, $\mathcal{P} = \{L_\theta : \mathcal{F}_\mathcal{X} \to [0,1], \theta \in \Omega\}$. Second, the measurements are generally available only for a finite set of times $\mathcal{T}$. To specify the random variables in question, we will need a notation for the event index at a given time $t$, $I(t) = \min\left\{N : \sum_{n=1}^{N+1} J_n > t\right\}$ (see Figure 1, where $I(t^*) = N - 1$), and for the individual observations, $Y(t)|X \sim L_{\theta_{I(t)}}$. The set of all observed random variable is then defined as $\mathcal{Y} = (Y(t_1), Y(t_2), \ldots, Y(t_G) : t_g < t_{g+1}, \{t_i\} = \mathcal{T})$.

For simplicity, we assume in this section that $\mathcal{P}$ is a conjugate family with respect to $H_0$. Non-conjugate models can be handled by incorporating the auxiliary variables of Algorithm 8 in [21]. We will describe inference on the model of Section 3. Extension to hierarchical models is direct (by keeping track of an additional sufficient statistic $G$, as well as the auxiliary variables $A_n, \|\mu_0\|$).

In general, there may be several exchangeable sequences from which we want to learn a model. For example, we learned a model for MS disease progression by using time series from several patients.[4] We denote the number of time series by $K$, each of the form

$$\mathcal{Y}^{(k)} = \left(Y^{(k)}(t_1^{(k)}), Y^{(k)}(t_2^{(k)}), \ldots, Y^{(k)}(t_G^{(k)}) : t_g^{(k)} < t_{g+1}^{(k)}, \{t_i^{(k)}\} = \mathcal{T}^{(k)}\right), \quad k \in \{1, \ldots, K\}.$$

At a high-level, our inference algorithm works by resampling the hidden events $X^{(k)}$ for one sequence $k$ given the sufficient statistics of the other sequences, $(F_\theta^{(\backslash k)}, T_\theta^{(\backslash k)})$. This is done using a Sequential Monte Carlo (SMC) algorithm to construct a proposal over sequences of hidden events. Each *particle* in our SMC algorithm is a sequence of states and waiting times for the current sequence $k$. By using a Particle MCMC (PMCMC) method [1], we then compute an acceptance ratio

| | Datasets | | | | Results (mean error) | | |
|---|---|---|---|---|---|---|---|
| Name | # sequences | # datapoints | # heldout | # characters | Baseline | EM | HGEP |
| Synthetic | 1000 | 10000 | 878 | 4 | 0.703 | **0.404** | 0.446 |
| MS | 72 | 384 | 31 | 3 | 0.516 | 0.355 | **0.277** |
| RNA | 1000 | 6167 | 508 | 4 | 0.648 | 0.596 | **0.426** |

Table 1: Summary statistics and mean error results for the experiments. All experiments were repeated 5 times.

that makes this proposal a valid MCMC move. As we will see shortly, the acceptance is simply given by a ratio of marginal likelihood estimators, which can be computed directly from the unnormalized particle weights.

Formally, the proposal is based on $M$ particles propagated from generation $g = 0$ up to generation $G$, where $G$ is equal to the number of measurements in the current sequence, $G = |\mathcal{Y}^{(k)}|$. Each particle $X_{m,g}$, $m \in \{1, \ldots, M\}$ consists of a list of hidden events indexed by $n$, containing both (hidden) states and waiting times: $X_{m,g} = (\theta_{m,n}, J_{m,n})_{n=1}^{N_{m,g}}$. The pseudocode for the SMC algorithm, used for constructing the proposals, is presented in Figure 4 of the Supplementary Material. The next step is to compute an acceptance probability for a proposed sequence of states $X_{*}^{(k)}$.

At each MCMC iteration, we assume that we store the value of the data likelihood estimates for the accepted state sequences. These data likelihood estimates are computed from the unnormalized weights $\pi_g$ (described in Figure 4 of the Supplementary Material) as follows: $L^{(k)} = \prod_{g=1}^{G} \|\pi_g\|$. Let $L^{(k)}$ be the estimate for the previously accepted sequence of states for observed sequence $k$, and let $L_{*}^{(k)}$ be the estimate for the current MCMC iteration. The acceptance probability for the new sequence is given by $\min\left\{1, L_{*}^{(k)}/L^{(k)}\right\}$. If it is accepted, we set $L^{(k)} = L_{*}^{(k)}$.

# 6 Experiments

In this section we present the results of our experiments. First, we demonstrate the behavior of state trajectories and sojourn times sampled from the prior to give a qualitative idea of the range of time series that can be captured by our model. Second, we evaluate quantitatively our model by applying it to three held-out tasks: synthetic, Multiple Sclerosis (MS) patients, and RNA evolutionary datasets.

## 6.1 Qualitative behavior of the prior

We can distinguish at least four types of prior behaviors in the HGEP when considering different values for the parameters $\beta_0$, $\|H_0\|$ and $\gamma_0$. We sampled a sequence of length $T = 800$ and present the state-time plots. Figure 2(a) shows a sequence with short sojourn times and high volatility of states, whereas Figure 2(b) depicts longer sojourn times with much less volatility. Figures 2(c) and 2(d) illustrate the effect of hyperparameter $\|H_0\|$. In Figure 2(c) we can see creation of many new states and a sparse transition matrix. Likewise, in Figure 2(d) the high tendency to create new states is present, but we have longer sojourn times. See Section H of the supplementary material for a more detailed account of the interpretation and quantitative effect of the parameters.

## 6.2 Quantitative evaluation

In this section, we use a simple likelihood model for discrete observations (described in Section G of the supplementary material) to evaluate our method on three held-out tasks. Note that even when the observations are discrete, non-parametric models are still useful for better explaining the data using latent variables [22].

We considered three evaluation datasets obtained by holding out each observed datapoint with a 10% probability (see Table 1). We then reconstructed the observations at these held-out times, and measured the mean error. For HGEP, reconstruction was done by using the Bayes estimator approximated from 1000 posterior samples (one after each scan through all the time series). We repeated

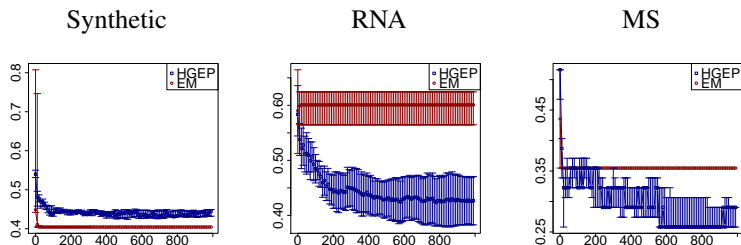

Figure 3: Mean reconstruction error on the held-out data as a function of the number of Gibbs scans. Lower is better. The standard maximum likelihood estimate learned with EM outperformed our model in the simple synthetic dataset, but the trend was reversed in the more complex real world datasets.

all experiments 5 times with different random seeds. We compared against the standard maximum likelihood rate matrix estimator learned by EM described in [23]. We also report in Table 1 the mean error for a simpler maximum likelihood estimate ignoring the sequential information (returning the most common observation deterministically). See Section G of the supplementary material for detailed instructions for replicating the following three results.[5] Refer also to Figure 3, where we show error as a function of the number of scans.

**Synthetic:** We used an Erdös-Rényi model to generate a random sparse matrix of size $10 \times 10$, which we perturbed with uniform noise to get a random rate matrix. Both HGEP and the EM-learned maximum likelihood outperformed the baseline. In contrast to the next two tasks, the EM approach slightly outperformed the HGEP model here. We believe this is because the synthetic data was not sufficiently rich to highlight the advantages of HGEPs. However, we compared our results with iHMM after discretizing time. We observed that iHMM had an error rate of 0.47, underperforming both EM and HGEP.

**MS disease progression:** This dataset, obtained from a phase III clinical trial, tracks the progression of MS in 72 patients over 3 years. The observed state of a patient at a given time is binned into three categories as customary in the MS literature [3]. Both HGEP and EM outperformed the baseline by a large margin, and our HGEP model outperformed EM with a relative error reduction of $22\%$.

**RNA evolution:** In this task, we used the dataset from [4] containing aligned 16S ribosomal RNA of species from the three domains of life. As a preprocessing, we constructed a rooted phylogenetic tree from a sample of 30 species, and performed ancestral reconstruction using a standard CTMC model and all the sampled taxa in the tree. We then considered the time series consisting of paths from one modern leaf to the root. The task is to reconstruct held-out nucleotides using only the data in this path. Again, both HGEP and EM outperformed the baseline, and our model outperformed EM with a relative error reduction of $29\%$.

## 7 Conclusion

We have introduced a method for non-parametric Bayesian modeling of recurrent, continuous time processes. The model has attractive properties and we show that the posterior computations can be done efficiently using a sampler based on particle MCMC methods. Most importantly, our experiments show that the model is useful for analyzing complex real world time series.

**Acknowledgments**

We would like to thank Arnaud Doucet, John Petkau and the anonymous reviewers for helpful comments. This work was supported by a NSERC Discovery Grant and the WestGrid cluster.

http://www.stat.ubc.ca/~bouchard/GEP/

## Footnotes

[1]The terminology "Moran Gamma Process" is from Kingman (e.g. in [17]). It is the same process as the Gamma process used in e.g. [15], except that we have one more degree of freedom in the parameterization (the rate; this is because ours is not destructively normalized).

[2]We use the rate parameterization for the gamma density throughout.

[3]Note that the GEP as defined above can generate self-transitions, but conditioning on the parameters, the jump waiting times are still exponential. However for computing predictive distributions, it will be simpler to allow positive self-transitions rates.

[4]Even in cases where there is a single long sequence, we recommend for efficiency reasons to partition the sequence into subsequences. In this case our proposal can be viewed as a block update.

[5]The code used to run these experiments is available at

# References

[1] C. Andrieu, A. Doucet, and R. Holenstein. Particle Markov chain Monte Carlo methods. *Journal Of The Royal Statistical Society Series B*, 2010.

[2] J. Van Gael, Y. Saatci, Y. W. Teh, and Z. Ghahramani. Beam sampling for the infinite hidden Markov model. In *ICML*, 2008.

[3] M. Mandel. Estimating disease progression using panel data. *Biostatistics*, 2010.

[4] J.J. Cannone, S. Subramanian, M.N. Schnare, J.R. Collett, L.M. D'Souza, Y. Du, B. Feng, N. Lin, L.V. Madabusi, K.M. Muller, N. Pande, Z. Shang, N. Yu, and R.R. Gutell. The comparative RNA web (CRW) site: An online database of comparative sequence and structure information for ribosomal, intron, and other RNAs. *BioMed Central Bioinformatics*, 2002.

[5] S.N. MacEachern. Dependent nonparametric processes. In *Section on Bayesian Statistical Science, American Statistical Association*, 1999.

[6] J.E. Griffin. The Ornstein-Uhlenbeck Dirichlet process and other time-varying processes for Bayesian nonparametric inference. *Journal of Statistical Planning and Inference*, 2008.

[7] J.E. Griffin and M.F.J. Steel. Stick-breaking autoregressive processes. *Journal of Econometrics*, 2011.

[8] M. F. J. Steel. *The New Palgrave Dictionary of Economics*, chapter Bayesian time series analysis. Palgrave Macmillan, 2008.

[9] S. Heiler. A survey on nonparametric time series analysis. CoFE Discussion Paper 99-05, Center of Finance and Econometrics, University of Konstanz, 1999.

[10] M. J. Beal, Z. Ghahramani, and C. E. Rasmussen. The infinite hidden Markov model. In *Machine Learning*. MIT Press, 2002.

[11] E.B. Fox, E.B. Sudderth, M.I. Jordan, and A.S. Willsky. An hdp-hmm for systems with state persistence. In *Proceedings of the International Conference on Machine Learning*, 2008.

[12] J. Van Gael, Y. W. Teh, and Z. Ghahramani. The infinite factorial hidden Markov model. In *NIPS'08*, 2008.

[13] P.A.P. Moran. *The Theory of Storage*. Methuen, 1959.

[14] M. Friesl. Estimation in the Koziol-Green model using a gamma process prior. *Austrian Journal of Statistics*, 2008.

[15] V. Rao and Y. W. Teh. Spatial normalized gamma processes. In *Advances in Neural Information Processing Systems*, 2009.

[16] L. Kuo and S. K. Ghosh. Bayesian nonparametric inference for nonhomogeneous Poisson processes. Technical report, University of Connecticut, Department of Statistics, 1997.

[17] J. F. C. Kingman. *Poisson Processes*. The Clarendon Press Oxford University Press, 1993.

[18] M. Schroder. Risk-neutral parameter shifts and derivatives pricing in discrete time. *The Journal of Finance*, 2004.

[19] D. Dufresne. G distributions and the beta-gamma algebra. *Electronic Journal of Probability*, 2010.

[20] Y. W. Teh, M. I. Jordan, M. J. Beal, and D. M. Blei. Hierarchical Dirichlet processes. *Journal of the American Statistical Association*, 2004.

[21] R. Neal. Markov chain sampling methods for Dirichlet process mixture models. Technical report, U of T, 2000.

[22] P. Liang, S. Petrov, M. I. Jordan, and D. Klein. The infinite PCFG using hierarchical Dirichlet processes. In *Empirical Methods in Natural Language Processing and Computational Natural Language Learning (EMNLP/CoNLL)*, 2007.

[23] A. Hobolth and J.L. Jensen. Statistical inference in evolutionary models of DNA sequences via the EM algorithm. *Statistical applications in Genetics and Molecular Biology*, 2005.

[24] L. Mateiu and B. Rannala. Inferring complex DNA substitution processes on phylogenies using uniformization and data augmentation. *Syst. Biol.*, 2006.

